# Interpolating Earth-science Data using RBF Networks and Mixtures of Experts

**E. Wan**        **D. Bone**
Division of Information Technology
Canberra Laboratory, CSIRO
GPO Box 664, Canberra, ACT, 2601, Australia
{ernest, don}@cbr.dit.csiro.au

## Abstract

We present a mixture of experts (ME) approach to interpolate sparse, spatially correlated earth-science data. Kriging is an interpolation method which uses a global covariation model estimated from the data to take account of the spatial dependence in the data. Based on the close relationship between kriging and the radial basis function (RBF) network (Wan & Bone, 1996), we use a mixture of generalized RBF networks to partition the input space into statistically correlated regions and learn the local covariation model of the data in each region. Applying the ME approach to simulated and real-world data, we show that it is able to achieve good partitioning of the input space, learn the local covariation models and improve generalization.

## 1. INTRODUCTION

Kriging is an interpolation method widely used in the earth sciences, which models the surface to be interpolated as a stationary random field (RF) and employs a linear model. The value at an unsampled location is evaluated as a weighted sum of the sparse, spatially correlated data points. The weights take account of the spatial correlation between the available data points and between the unknown points and the available data points. The spatial dependence is specified in the form of a global covariation model. Assuming global stationarity, the kriging predictor is the best unbiased linear predictor of the unsampled value when the true covariation model is used, in the sense that it minimizes the squared error variance under the unbiasedness constraint. However, in practice, the covariation of the data is unknown and has to be estimated from the data by an initial spatial data analysis. The analysis fits a covariation model to a covariation measure of the data such as the sample variogram or the sample covariogram, either graphically or by means of various least squares (LS) and maximum likelihood (ML) approaches. Valid covariation models are all radial basis functions.

Optimal prediction is achieved when the true covariation model of the data is used. In general, prediction (or generalization) improves as the covariation model used more

closely matches the true covariation of the data. Nevertheless, estimating the covariation model from earth-science data has proved to be difficult in practice due to the sparseness of data samples. Furthermore for many data sets the global stationarity assumption is not valid. To address this, data sets are commonly manually partitioned into smaller regions within which the stationarity assumption is valid or approximately so.

In a previous paper, we showed that there is a close, formal relationship between kriging and RBF networks (Wan & Bone, 1996). In the equivalent RBF network formulation of kriging, the input vector is a coordinate and the output is a scalar physical quantity of interest. We pointed out that, under the stationarity assumption, the radial basis function used in an RBF network can be viewed as a covariation model of the data. We showed that an RBF network whose RBF units share an adaptive norm weighting matrix, can be used to estimate the parameters of the postulated covariation model, outperforming more conventional methods. In the rest of this paper we will refer to such a generalization of the RBF network as a generalized RBF (GRBF) network.

In this paper, we discuss how a mixture of GRBF networks can be used to partition the input space into statistically correlated regions and learn the local covariation model of each region. We demonstrate the effectiveness of the ME approach with a simulated data set and an aero-magnetic data set. Comparisons are also made of prediction accuracy of a single GRBF network and other more traditional RBF networks.

## 2   MIXTURE OF GRBF EXPERTS

Mixture of experts (Jacobs et al , 1991) is a modular neural network architecture in which a number of expert networks augmented by a gating network compete to learn the data. The gating network learns to assign probability to the experts according to their performance over various parts of the input space, and combines the outputs of the experts accordingly. During training, each expert is made to focus on modelling the local mapping it performs best, improving its performance further. Competition among the experts achieves a soft partitioning of the input space into regions with each expert network learning a separate local mapping. An hierarchical generalization of ME, the hierarchical mixture of experts (HME), in which each expert is allowed to expand into a gating network and a set of sub-experts, has also been proposed (Jordan & Jacobs, 1994).

Under the global stationarity assumption, training a GRBF network by minimizing the mean squared prediction error involves adjusting its norm weighting matrix. This can be interpreted as an attempt to match the RBF to the covariation of the data. It then seems natural to use a mixture of GRBF networks when only local stationarity can be assumed. After training, the gating network soft partitions the input space into statistically correlated regions and each GRBF network provides a model of the covariation of the data for a local region. Instead of an ME architecture, an HME architecture can be used. However, to simplify the discussion we restrict ourselves to the ME architecture.

Each expert in the mixture is a GRBF network. The output of expert $i$ is given by:

$$\hat{y}_i(\mathbf{x};\boldsymbol{\theta}_i) = \sum_{j=1}^{n_i} w_{ij}\phi(\mathbf{x};\mathbf{c}_{ij},\mathbf{M}_i) + w_{i0} \qquad (2.1)$$

where $n_i$ is the number of RBF units, $\boldsymbol{\theta}_i = \{\{w_{ij}\}_{j=0}^{n_i},\{\mathbf{c}_{ij}\}_{j=1}^{n_i},\mathbf{M}_i\}$ are the parameters of the expert and $\phi(\mathbf{x};\mathbf{c},\mathbf{M})=\varphi(\|\mathbf{x}\text{-}\mathbf{c}\|_M)$. Assuming zero-mean Gaussian error and common variance $\sigma_i^2$, the conditional probability of $y$ given $\mathbf{x}$ and $\theta_i$ is given by:

$$P(y|\mathbf{x},\boldsymbol{\theta}_i) = \tfrac{1}{\sqrt{2\pi}\sigma_i}\exp\left(-\tfrac{1}{2\sigma_i^2}(y-\hat{y}_i(\mathbf{x};\boldsymbol{\theta}_i))^2\right). \qquad (2.3)$$

Since the radial basis functions we used have compact support and each expert only learns a local covariation model, small GRBF networks spanning overlapping regions can be used to reduce computation at the expense of some resolution in locating the boundaries of the regions. Also, only the subset of data within and around the region spanned by a GRBF network is needed to train it, further reducing computational effort.

With $m$ experts, the $i^{th}$ output of the gating network gives the probability of selecting the expert $i$ and is given by the normalized function:

$$g_i(\mathbf{x};\upsilon) = P(i|\mathbf{x},\upsilon) = \alpha_i \exp(q(\mathbf{x};\upsilon_i)) \Big/ \sum_{j=1}^{m} \alpha_j \exp(q(\mathbf{x};\upsilon_j)) \qquad (2.4)$$

where $\upsilon = \{\{\alpha_i\}_{i=1}^{m}, \{\upsilon_i\}_{i=1}^{m}\}$. Using $q(\mathbf{x};\upsilon_i) = \upsilon_i^T[\mathbf{x}^T \ 1]^T$ and setting all $\alpha_i$'s to 1, the gating network implements the softmax function and partitions the input space into a smoothed planar tessellation. Alternatively, with $q(\mathbf{x};\upsilon_i) = -\|\mathbf{T}_i(\mathbf{x}-\mathbf{u}_i)\|^2$ (where $\upsilon_i=\{\mathbf{u}_i, \mathbf{T}_i\}$ consists of a location vector and an affine transformation matrix) and restricting the $\alpha_i$'s to be non-negative, the gating network divides the input space into packed anisotropic ellipsoids. These two partitionings are quite convenient and adequate for most earth-science applications where $\mathbf{x}$ is a 2D or 3D coordinate.

The output of the experts are combined to give the overall output of the mixture:

$$\hat{y}(\mathbf{x};\theta) = \sum_{i=1}^{m} P(i|\mathbf{x},\upsilon)\hat{y}_i(\mathbf{x};\theta_i) = \sum_{i=1}^{m} g_i(\mathbf{x};\upsilon)\hat{y}_i(\mathbf{x};\theta_i) \qquad (2.5)$$

where $\theta = \{\upsilon, \{\theta_i\}_{i=1}^{m}\}$ and the conditional probability of observing $y$ given $\mathbf{x}$ and $\theta$ is:

$$P(y|\mathbf{x},\theta) = \sum_{i=1}^{m} P(i|\mathbf{x},\upsilon)P(y|\mathbf{x},\theta_i). \qquad (2.6)$$

## 3   THE TRAINING ALGORITHM

The Expectation-Maximization (EM) algorithm of Jordan and Jacobs is used to train the mixture of GRBF networks. Instead of computing the ML estimates, we extend the algorithm by including priors on the parameters of the experts and compute the maximum a posteriori (MAP) estimates. Since an expert may be focusing on a small subset of the data, the priors help to prevent over-fitting and improve generalization.

Jordan & Jacobs introduced a set of indicator random variables $Z = \{z^{(t)}\}_{t=1}^{N}$ as missing data to label the experts that generate the observable data $D = \{(\mathbf{x}^{(t)}, y^{(t)})\}_{t=1}^{N}$. The log joint probability of the complete data $D_c = \{D, Z\}$ and parameters $\theta$ can be written as:

$$\ln P(D_c, \theta|\lambda) = \ln\left\{ P(\theta|\lambda)\prod_{t=1}^{N}\prod_{i=1}^{m}\left\{P(i|\mathbf{x}^{(t)},\upsilon)P(y^{(t)}|\mathbf{x}^{(t)},\theta_i)\right\}^{z_i^{(t)}}\right\} \qquad (3.1)$$

where $\lambda$ is a set of hyperparameters. Assuming separable priors on the parameters of the model i.e. $P(\theta|\lambda) = P(\upsilon|\lambda_0)\prod_{i=1}^{m} P(\theta_i|\lambda_i)$ with $\lambda = \{\lambda_i\}_{i=0}^{m}$, (3.1) can be rewritten as:

$$\begin{aligned}
\ln P(D_c, \theta|\lambda) &= \sum_{t=1}^{N}\sum_{i=1}^{m} z_i^{(t)} \ln P(i|\mathbf{x}^{(t)},\upsilon) + \ln P(\upsilon|\lambda_0) \\
&\quad + \sum_{i=1}^{m}\left\{\sum_{t=1}^{N} z_i^{(t)} \ln P(y^{(t)}|\mathbf{x}^{(t)},\theta_i) + \ln P(\theta_i|\lambda_i)\right\}
\end{aligned} \qquad (3.2)$$

Since the posterior probability of the model parameters is proportional to the joint probability, maximizing (3.2) is equivalent to maximizing the log posterior. In the E-step, the observed data and the current network parameters are used to compute the expected value of the complete-data log joint probability:

$$Q\left(\theta|\theta^{(k)}\right) = \sum_{t=1}^{N}\sum_{i=1}^{m} h_i^{(k)}(t)\ln P\left(i|\mathbf{x}^{(t)},\upsilon\right) + \ln P\left(\upsilon|\lambda_0\right)$$
$$+ \sum_{i=1}^{m}\left\{\sum_{t=1}^{N} h_i^{(k)}(t)\ln P\left(y^{(t)}|\mathbf{x}^{(t)},\theta_i\right) + \ln P\left(\theta_i|\lambda_i\right)\right\} \qquad (3.3)$$

where
$$h_i^{(k)}(t) = E\left[z_i^{(t)}|D,\theta^{(k)}\right] = P\left(i|\mathbf{x}^{(t)},y^{(t)}\right) = \frac{P\left(i|\mathbf{x}^{(t)},\upsilon^{(k)}\right)P\left(y^{(t)}|\mathbf{x}^{(t)},\theta_i^{(k)}\right)}{\sum_{j=1}^{m} P\left(j|\mathbf{x}^{(t)},\upsilon^{(k)}\right)P\left(y^{(t)}|\mathbf{x}^{(t)},\theta_j^{(k)}\right)} \qquad (3.4)$$

In the M-step, $Q(\theta|\theta^{(k)})$ is maximized with respect to $\theta$ to obtain $\theta^{(k+1)}$. As a result of the use of the indicator variables, the problem is decoupled into a separate set of interim MAP estimations:

$$\upsilon^{(k+1)} = \arg\max_{\upsilon} \sum_{t=1}^{N}\sum_{i=1}^{m} h_i^{(k)}(t)\ln P\left(i|\mathbf{x}^{(t)},\upsilon\right) + \ln P\left(\upsilon|\lambda_0\right) \qquad (3.5)$$

$$\theta_i^{(k+1)} = \arg\max_{\theta_i} \sum_{t=1}^{N} h_i^{(k)}(t)\ln P\left(y^{(t)}|\mathbf{x}^{(t)},\theta_i\right) + \ln P\left(\theta_i|\lambda_i\right) \qquad (3.6)$$

We assume a flat prior for the gating network parameters and the prior $P(\theta_i|\lambda_i) = \exp(-\frac{1}{2}\lambda_i \sum_{r=1}^{n_i}\sum_{s=1}^{n_i} w_{ir}w_{is}\phi(\mathbf{c}_{ir}-\mathbf{c}_{is}))/Z_R(\lambda_i)$ where $Z_R(\lambda_i)$ is a normalization constant, for the experts. This smoothness prior is used on the GRBF networks because it can be derived from regularization theory (Girosi & Poggio, 1990) and at the same time is consistent with the interpretation of the radial basis function as a covariation model. Hence, maximizing $\theta_i$ with (3.6) is equivalent to minimizing the cost function:

$$E_i = \sum_{t=1}^{N} h_i^{(k)}(t)\left(y\left(\mathbf{x}_j\right) - \hat{y}_i\left(\mathbf{x}_j\right)\right)^2 + \lambda_i^* \sum_{r=1}^{n_i}\sum_{s=1}^{n_i} w_{ir}w_{is}\phi\left(\mathbf{c}_{ir}-\mathbf{c}_{is}\right) \qquad (3.7)$$

where $\lambda_i^* = \lambda_i\sigma_i^2$. The value of the effective regularization parameter, $\lambda_i^*$, can be set by generalized cross validation (GCV) (Orr, 1995) or by the 'evidence' method of (Mackay, 1991) using re-estimation formulas. However, in the simulations, for simplicity, we preset the value of the regularization parameter to a fixed value.

## 4   SIMULATION RESULTS

Using the Cholesky decomposition method (Cressie, 1993), we generate four 2D data sets using the four different covariation models shown in Figure 1. The four data set are then joined together to form a single 64x64 data set. Figure 3a shows the original data set and the hard boundaries of the 4 statistically distinct regions. We randomly sample the data to obtain a 400 sample training set and use the rest of the data for validation.

Two GRBF networks, with 64 and 144 adaptive anistropic spherical[1] units respectively, are used to learn the postulated global covariation model and the mapping. A 2-level

HME with 4 GRBF network experts each with 36 spherical units are used to learn the local covariation models and the mapping. Softmax gating networks are used and each expert is somewhat 'localized' in each quadrant of the input space. The units of the experts are located at the same locations as the units of the 64-unit GRBF network with 24 overlapping units between any two of the experts. The design ensures that the HME does not have an advantage over the 64-unit GRBF network if the data is indeed globally stationary. Figure 2 shows the local covariation models learned by the HME with the smoothness priors and Figure 3b shows the interpolant generated and the partitioning.

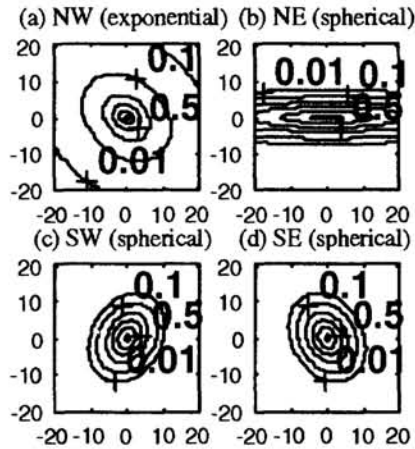

Figure 1: The profile of the true local covariation models of the simulated data set. Exponential and spherical models are used.

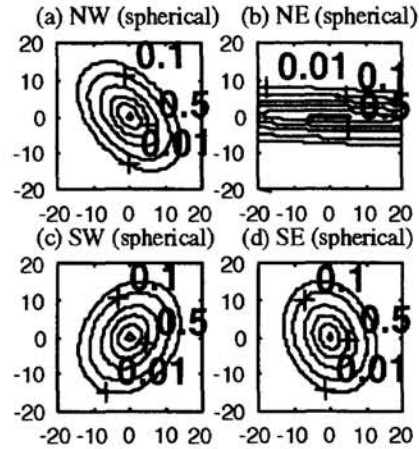

Figure 2: The profile of the local covariation models learned by the HME.

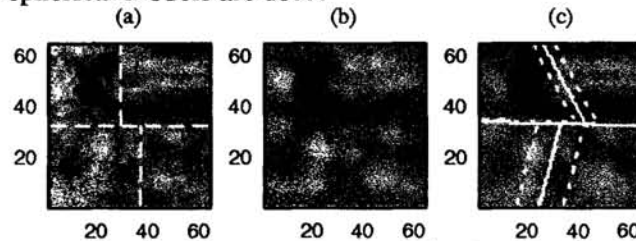

Figure 3: (a) Simulated data set and true partitions. (b) Interpolant generated by the 144 spherical unit GRBFN. (c) The HME interpolant and the soft partitioning learned (0.5, 0.9 probability contours of the 4 experts shown in solid and dotted lines respectively)

Table 1: Normalized mean squared prediction error for the simulated data set.

| Network | RBF unit | NMSE |
|---|---|---|
| RBFN *(isotropic RBF units with width set to the distance to the nearest neighbor)* | 64, Gaussian | 0.761 |
| | 144, Gaussian | 0.616 |
| | 400, Gaussian | 0.543 |
| RBFN *(identical isotropic RBF units with adaptive width)* | 64, Gaussian | 0.477 |
| | 144, Gaussian | 0.475 |
| GRBFN *(identical RBF units with adaptive norm weighting matrix)* | 64, spherical | 0.506 |
| | 144, spherical | 0.431 |
| HME *(2 levels, 4 GRBFN experts) without priors* | 4x36, spherical | 0.938 |
| HME *(2 levels, 4 GRBFN experts) with priors* | 4x36, spherical | 0.433 |
| kriging predictor *(using true local models)* | | 0.372 |

For comparison, a number of ordinary RBF networks are also used to learn the mapping. In all cases, the RBF units of networks of the same size share the same locations which

are preset by a Kohonen map. Table 1 summarizes the normalized mean squared prediction error (NMSE)- the squared prediction error divided by the variance of the validation set - for each network. With the exception of HME, all results listed are obtained with a smoothness prior and a regularization parameter of 0.1. Ordinary weight decay is used for RBF networks with units of varying widths and the smoothness prior discussed in section 3 are used for the remaining networks. The NMSE of the kriging predictor that uses the true local models is also listed as a reference.

Similar experiments are also conducted on a real aero-magnetic data set. The flight paths along which the data is collected are divided into a 740 data points training set and a 1690 points validation set. The NMSE for each network is summarized in Table 2, the local covariation models learned by the HME is shown in Figure 4, and the interpolant generated by the HME and the partitioning is shown in Figure 5b.

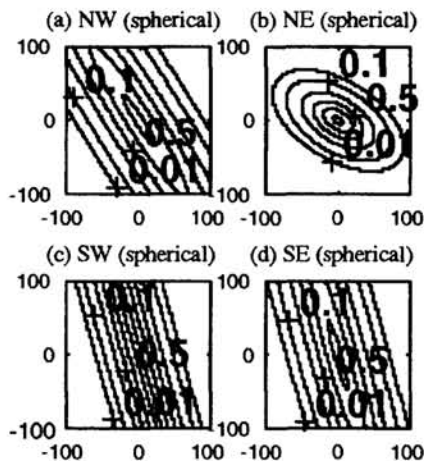

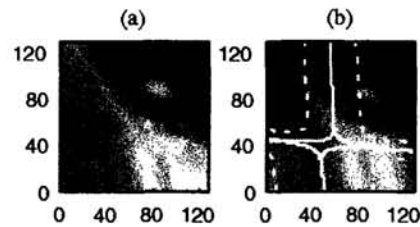

Figure 5: (a) Thin-plate interpolant of the entire aero-magnetic data set. (b) The HME interpolant and the soft partitioning (0.5, 0.9 probability contours of the 4 experts shown in solid and dotted lines respectively).

Figure 4: The profile of the local covariation models of the aero-magnetic data set learned by the HME.

Table 2: Normalized mean squared prediction error for the aero-magnetic data set.

| Network | RBF units | NMSE |
|---|---|---|
| RBFN *(isotropic RBF units with width set to the* | 49, Gaussian | 1.158 |
| *distance to the nearest neighbor)* | 100, Gaussian | 1.256 |
| RBFN *(isotropic RBF units with width set to the* | 49, Gaussian | 0.723 |
| *mean distance to the 8 nearest neighbors)* | 100, Gaussian | 0.699 |
| RBFN *(identical isotropic RBF units with adaptive* | 49, Gaussian | 0.692 |
| *width)* | 100, Gaussian | 0.614 |
| GRBFN *(identical RBF units with adaptive norm* | 49, spherical | 0.684 |
| *weighting matrix)* | 100, spherical | 0.612 |
| HME *(2 levels, 4 GRBFN experts) without priors* | 4x25, spherical | 0.389 |
| HME *(2 levels, 4 GRBFN experts) with priors* | 4x25, spherical | 0.315 |

# 5 DISCUSSION

The ordinary RBF networks perform worst with both the simulated data and the aero-magnetic data. As neither data set is globally stationary, the GRBF networks do not improve prediction accuracy over the corresponding RBF networks that use isotropic Gaussian units. In both cases, the hierarchical mixture of GRBF networks improves the prediction accuracy when the smoothness priors are used. Without the priors, the ML estimates of the HME parameters lead to improbably high and low predictions.

The improvement in prediction accuracy is more significant for the aero-magnetic data set than for the simulated data set due to some apparent global covariation of the simulated data which only becomes evident when the directional variograms of the data are plotted. However, despite the similar NMSE, Figure 3 shows that the interpolant generated by the 144-unit GRBF network does not contain the structural information that is captured by the HME interpolant and is most evident in the north-east region.

In the case of the simulated data set, the HME learns the local covariation models accurately despite the fact that the bottom level gating networks fail to partition the input space precisely along the north-south direction. The availability of more data and the straight east-west discontinuity allows the upper gating network to partition the input space precisely along the east-west direction. In the north-west region, although the class of function the expert used is different from that of the true model, the model learned still resembles the true model especially in the inner region where it matters most.

In the case of the aero-magnetic data set, the RBF and GRBF networks perform poorly due to the considerable extrapolation that is required in the prediction and the absence of global stationarity. However, the HME whose units capture the local covariation of the data interpolates and extrapolates significantly better. The partitioning as well as the local covariation model learned by the HME seems to be reasonably accurate and leads to the construction of prominent ridge-like structures in the north-west and south-east which are only apparent in the thin-plate interpolant of the entire data set of Figure 5a.

## 6 CONCLUSIONS

We show that a mixture of GRBF networks can be used to learn the local covariation of spatial data and improve prediction (or generalization) when the data is approximately locally stationary - a viable assumption in many earth-science applications. We believe that the improvement will be even more significant for data sets with larger spatial extent especially if the local regions are more statistically distinct. The estimation of the local covariation models of the data and the use of these models in producing the interpolant helps to capture the structural information in the data which, apart from accuracy of the prediction, is of critical importance to many earth-science applications.

The ME approach allows the objective and automatic partitioning of the input space into statistically correlated regions. It also allows the use of a number of small local GRBF networks each trained on a subset of the data making it scaleable to large data sets.

The mixture of GRBF networks approach is motivated by the statistical interpolation method of kriging. The approach therefore has a very sound physical interpretation and all the parameters of the network have clear statistical and/or physical meanings.

## Footnotes

[1] The spherical model is widely used in geostatistics and when used as a covariance function is defined as $\varphi(\mathbf{h};a) = 1 - \{\frac{3}{2}(\frac{\|\mathbf{h}\|}{a}) - \frac{1}{2}(\frac{\|\mathbf{h}\|}{a})^3\}$ for $0 \le \|\mathbf{h}\| \le a$ and $\varphi(\mathbf{h};a) = 0$ for $\|\mathbf{h}\| > a$. Spherical does NOT mean isotropic.

## References

Cressie, N. A. (1993). *Statistics for Spatial Data*. Wiley, New York.
Jacobs, R. A., Jordan, M. I., Nowlan, S. J. & Hinton, G. E. (1991). Adaptive Mixtures of Local Experts. *Neural Computation* 3, pp. 79-87.
Jordan, M. I. & Jacobs, R. A. (1994). Hierarchical Mixtures of Experts and the EM Algorithm. *Neural Computation* 6, pp. 181-214.
MacKay, D. J. (1992). Bayesian Interpolation. *Neural Computation* 4, pp. 415-447.
Orr, M. J. (1995). Regularization in the Selection of Radial Basis Function Centers. *Neural Computation* 7, pp. 606-623.
Poggio, T. & Girosi, F. (1990). Networks for Approximation and Learning. In *Proceedings of the IEEE* 78, pp. 1481-1497.
Wan, E. & Bone, D. (1996). A Neural Network Approach to Covariation Model Fitting and the Interpolation of Sparse Earth-science Data. In *Proceedings of the Seventh Australian Conference on Neural Networks*, pp. 121-126.